# Competence Acquisition in an Autonomous Mobile Robot using Hardware Neural Techniques.

**Geoff Jackson and Alan F. Murray**
Department of Electrical Engineering
Edinburgh University
Edinburgh, EH9 3JL
Scotland, UK
gbj@ee.ed.ac.uk,afm@ee.ed.ac.uk

## Abstract

In this paper we examine the practical use of hardware neural networks in an autonomous mobile robot. We have developed a hardware neural system based around a custom VLSI chip, EP-SILON II[1], designed specifically for embedded hardware neural applications. We present here a demonstration application of an autonomous mobile robot that highlights the flexibility of this system. This robot gains basic mobility competence in very few training epochs using an "instinct-rule" training methodology.

## 1 INTRODUCTION

Though neural networks have been shown as an effective solution for a diverse range of real-world problems, applications and especially hardware implementations have been few and slow to emerge. For example in the DARPA neural networks study of 1988; of the 77 neural network applications investigated only 4 had resulted in field tested systems [Widrow, 1988]. Furthermore, none of these used dedicated neural network hardware. It is our view that this lack of tangible successes can be summarised by the following points:

- Most neural applications will be served optimally by fast, generic digital computers.

- Dedicated digital neural accelerators have a limited lifetime as "the fastest", as standard computers develop so rapidly.

- Analog neural VLSI is a niche technology, optimally applied at the interface between the real world and higher-level digital processing.

This attitude has some profound implications with respect to the size, nature and constraints we place on new hardware neural designs. After several years of research into hardware neural network implementation, we have now concentrated on the areas in which analog neural network technology has an "edge" over well established digital technology.

Within the pulse stream neural network research at the University of Edinburgh, the EPSILON chip's areas of strength can be summarised as:

- Analog or digital inputs, digital outputs.
- Scaleable and cascadeable design.
- Modest size.
- Compact, low power.

This list points naturally and strongly to problems on the boundary of the real, analog world and digital processing, such as pre-processing/interpretation of analog sensor data. Here a modest neural network can act as an *intelligent analog-to-digital converter* presenting preprocessed information to its host. We are now engaged in a two pronged approach, whereby development of technology to improve the performance of pulse stream neural network chips is occurring concurrently with a search and development of applications to which this technology can be applied. The key requirements of this technological development are that devices must:

- Work directly with analog signals.
- Provide a moderate size network.
- Have the potential for a fully integrated solution.

In working with the above constraints and goals we have developed a new chip, EPSILON II, and a bus based processor card incorporating it. It is our aim to use this system to develop applications. As our first demonstration the EPSILON processor card has been mounted on an autonomous mobile robot. In this case the network utilises a mixture of analog and digital sensor information and performs a mapping between input/sensor space, a mixture of analog and digital signals, and output motor control.

## 2   THE EPSILON II CHIP

The EPSILON II chip has been designed around the requirements of an application based system. It follows on from an earlier generation of pulse stream neural network chip, the EPSILON chip [Murray, 1992].

The EPSILON II chip represents neural states as a pulse encoded signal. These pulse encoded signals have digital signal levels which make them highly immune to noise and ideal for inter and intra-chip communication, facilitating efficient cascading of chips to form larger systems. The EPSILON II chip can take as inputs either pulse encoded signals or analog voltage levels, thus facilitating the fusing of analog and digital data in one system. Internally the chip is analog in nature allowing the synaptic multiplication function to be carried out in compact and efficient analog cells [Jackson, 1994].

Table 1 shows the principal specifications of the EPSILON II chip. The EPSILON II chip is based around a 32x32 synaptic matrix allowing efficient interfacing to digital systems. Several features of the device have been developed specifically for applications based usage. The first of these is a programmable input mode. This

Table 1: EPSILON II Specifications

| EPSILON II Chip Specifications | |
|---|---|
| No. of state input pins | 32 |
| Input modes | Analog, PW or PF |
| Input mode programmability | Bit programmable |
| No. of state outputs | 32 pinned out |
| Output modes | PW or PF |
| Digital recovery of analog I/P | Yes - PW encoded |
| No. of Synapses | 1024 |
| Additional *autobias* synapses | 4 per output neuron |
| Weight storage | Dynamic |
| Programmable activity voltage | Yes |
| Die size | $6.9mm \times 7mm$ |

allows each of the network inputs to be programmed as either a direct analog input or a digital pulse encoded input. We believe that this is vital for application based usage where it is often necessary to *fuse* real–world analog data with historical or control data generated digitally. The second major feature is a pulse recovery mode. This allows conversion of any analog input into a digital value for direct use by the host system. Both these features are utilised in the robotics application described in section 4 of this paper.

## 3   EPSILON PROCESSOR CARD

The need to embed the EPSILON chip in a processor card is driven by several considerations. Firstly, working with pulse encoded signals requires substantial processing to interface directly to digital systems. If the neural processor is to be transparent to the host system and is not to become a substantial processing overhead, then all pulse support operations must be carried out independently of the host system. Secondly, to respond to further chip level advances and allow rapid prototyping of new applications as they emerge, a certain amount of flexibility is needed in the system. It is with these points in mind that the design of the flexible EPSILON Processor Card (EPC) was undertaken.

### 3.1   DESIGN SPECIFICATION

The EPC has been designed to meet the following specifications. The card must:

- Operate on a conventional digital bus system.
- Be transparent to the host processor, that is carry out all the necessary pulse encoding and decoding.
- Carry out the refresh operations of the dynamic weights stored on the EPSILON chip.
- Generate the ramp waveforms necessary for pulse width coding.
- Support the operation of multiple EPC's.
- Allow direct input of analog signals.

As all data used and generated by the chip is effectively of 8-bit resolution, the STE bus, an industry standard 8-bit bus, was chosen for the bus system. This is also cost

effective and allows the use of readily available support cards such as processors, DSP cards and analog and digital signal conditioning cards.

To allow the transparency of operation the card must perform a variety of functions. A block diagram indicating these functions is shown in figure 1.

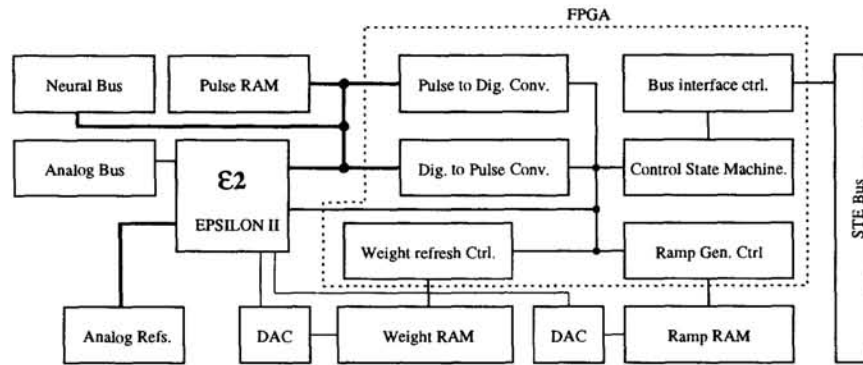

Figure 1: EPSILON Processor Card

A substantial amount of digital processing is required by the card, especially in the pulse conversion circuitry. To conform to the *Eurocard* standard size of the STE specification an FPGA device is used to "absorb" most of the digital logic. A twin mother/daughter board design is also used to isolate sensitive analog circuitry from the digital logic. The use of the FPGA makes the card extremely versatile as it is now easily reconfigurable to adapt to specialist application. The dotted box of figure 1 shows functions implemented by the FPGA device. An on board EPROM can hold multiple FPGA configurations such that the board can be reconfigured "on the fly". All EPSILON support functions, such as ramp generation, weight refresh, pulse conversion and interface control are carried out on the card. Also the use of the FPGA means that new ideas are easily tested as all digital signal paths go via this device. Thus a card of new functionality can be designed without the need to design a new PCB.

## 3.2 SPECIALIST BUSES

The digital pulse bus is buffered out under control of the FPGA to the neural bus along with two control signals. Handshaking between EPC's is done over these lines to allow the transfer of pulse stream data between processors. This implies that larger networks can be implemented with little or no increase in computation time or overhead. A separate analog bus is included to bring analog inputs directly onto the chip.

## 4   APPLICATIONS DEVELOPMENT

The over-riding reason for the development of the EPC is to allow the easy development of hardware neural network applications. We have already indicated that we believe that this form of neural technology will find its niche where its advantages of direct sensor interface, compactness and cost-effectiveness are of prime importance. As a good and intrinsically interesting example of this genre of applications, we have chosen autonomous mobile robotic control as a first test for EPSILON II. The object of this demonstrator is not to advance the state-of-the-art in robotics.

Rather it is to demonstrate analog neural VLSI in an appropriate and stimulating context.

## 4.1 "INSTINCT-RULE" ROBOT

The "instinct-rule" robotic control philosophy is based on a software-controlled exemplar from the University's Department of Artificial Intelligence [Nehmzow, 1992]. The robot incorporates an EPC which interfaces all the analog sensor signals and provides the programmable neural link between sensor/input space and the motor drive actuators.

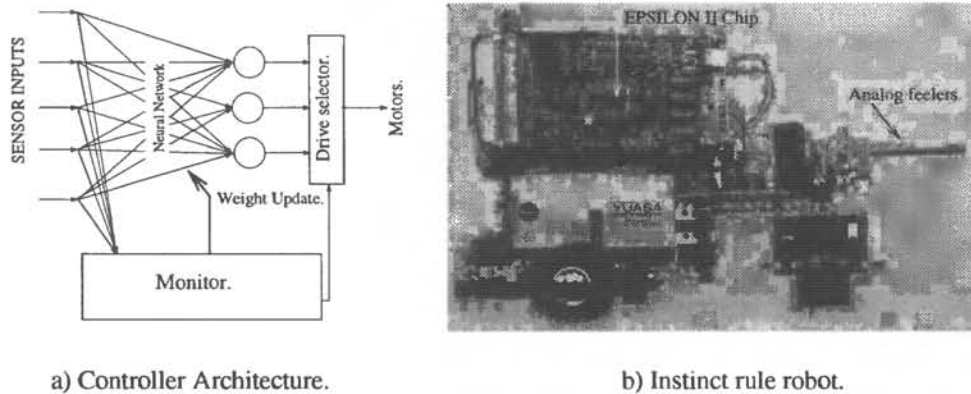

a) Controller Architecture.                    b) Instinct rule robot.

Figure 2: "Instinct Rule" Robot

The controller architecture is shown in figure 2. The neural network implemented on the EPC is the *plastic* element that determines the mapping between sensory data and motor actions. The majority of the monitor section is currently implemented on a host processor and monitors the performance of the neural network. It does this by regularly evaluating a set of *instinct rules*. These rules are simple behaviour based axioms. For example, we use two rules to promote simple obstacle avoidance competence in the robot, as listed in column one of table 2

Table 2: Instinct Rules

| Simple obstacle avoidance. | | Wall following | |
|---|---|---|---|
| 1. | **Keep crash sensors inactive.** | 1. | **Keep crash sensors inactive.** |
| 2. | **Move forward.** | 2. | **Keep side sensors active.** |
| | | 3. | **Move forward.** |

If an instinct rule is violated the drive selector then chooses the next strongest output (motor action) from the neural network. This action is then performed to see if it relieves the violation. If it does, it is used as targets to train the neural network. If it does not, the next strongest action is tried. The mechanism to accomplish this will be described in more detail in section 4.2.

Using this scheme the robot can be initialised with random weights (i.e. no mapping between sensors and motor control) and within a few epochs obtains basic obstacle avoidance competence.

It is a relatively easy matter to promote more complex behaviour with the addition of other rules. For example to achieve a wall following behaviour a third

rule is introduced as shown in column two of table 2. Navigational tasks can be accomplished with the addition of a **"maximise navigational signal"** rule. An example of this is a light sensor mounted on the robot producing a behaviour to move towards a light source. Equally, a signal from a more complex, higher level, navigational system could be used. Thus the instinct rule controller handles basic obstacle avoidance competence and motor/sensory interface tasks leaving other resources free for intensive navigational tasks.

## 4.2  INSTINCT RULE EVALUATION USING SOMATIC TENSION

The original instinct rule robot used binary sensor signals and evaluated performance of alternative actions for fixed, and progressively longer, periods of time [Nehmzow, 1992]. With the EPC interfacing directly to analog sensors an improved scheme has been developed. If we sum all sensors onto a neuron with fixed and equal weights we gain a measure of total sensory activity. Let us call this *somatic tension* as an analogy to biological signal aggregation on the soma. If we have an instinct violation and an alternative action is performed we can monitor this somatic tension to gauge the performance of this action. If tension decreases significantly we continue the action. If it increases significantly we choose an alternative action. If tension remains high and roughly the same, we are in a *tight* situation, for example say a corner. In this case we perform actions for progressively longer periods continuing to monitor somatic tension for a drop.

## 4.3  RESULTS AND DISCUSSION

The instinct rule robot has been constructed and its performance is comparable with software-controlled predecessors. Unfortunately direct comparisons are not possible due to unavailability of the original exemplars and differing physical characteristics of the robots themselves. In developing the application several observations were made concerning the behaviour of the system that would not have come to light in a simulated environment.

In any system including real mechanics and real analog signals, imperfections and noise are present. For example, in a real robot we cannot guarantee that a forward motion directive will result in perfect forward motion due to inherent asymmetries in the system. The instinct rule architecture does not assume a-priori knowledge such as this so behaviour is not affected adversely. This was tested by retarding one drive motor of the robot to give it a bias to one side.

In early development, as the monitor was being *tuned*, the robot showed a tendency to oscillatory motion, thus exhibiting undesirable behaviour that satisfies its instincts. It could, for example, oscillate back and forth at a corner. In a simulated environment this continues indefinitely. However, with real mechanics and noisy analog sensors the robot breaks out of this undesirable behaviour.

These observations strengthen the arguments for hardware development aimed at embedded systems. The robot application is but an example of the different, and often surprising conditions that pertain in a "real" system. If neural networks are to find applications in real-world, low-cost and analog-interface applications, these are the conditions we must deal with, and appropriate, analog hardware is the optimal medium for a solution.

# 5   CONCLUSIONS

This paper has described pulse stream neural networks that have been developed to a system level to aid development of applications. We have therefore defined areas of strengths of this technology along with suggestions of where this is best applied. The strengths of this system include:

1. Direct interfacing to analog signals.
2. The ability to fuse direct analog sensor data with digital sensor data processed elsewhere in the system.
3. Distributed processing. Several EPC's may be embedded in a system to allow multiple networks and/or multi layer networks.
4. The EPC represents a flexible system level development environment. It is easily reconfigured for new applications or improved chip technology.
5. The EPC requires very little computational overhead from the host system and can operate independently if needed.

A demonstration application of an instinct rule robot has been presented highlighting the use of neural networks as an interface between real-world analog signals and digital control.

In conclusion we believe that the immediate future of neural analog VLSI is in small applications based systems that interface directly to the real-world. We see this as the primary niche area where analog VLSI neural networks will replace conventional digital systems.

## Acknowledgements

Thanks are due to Ulrich Nehmzow, University of Manchester, for discussions and information on the instinct-rule controller and the loan of his original robot – Alder.

## Footnotes

[1]Edinburgh Pulse Stream Implemenation of a Learning Oriented Network.

## References

[Caudell, 1990] Caudell, M. and Butler, C. (1990). *Naturally Intelligent Systems.* MIT Press, Cambridge, Ma.

[Jackson, 1994] Jackson, G., Hamilton, A., and Murray, A. F. (1994). Pulse stream VLSI neural systems: into robotics. In *Proceedings ISCAS'94*, volume 6, pages 375–378. IEEE Press.

[Maren, 1990] Maren, A., Harston, C., and Pap, R. (1990). *Handbook of Neural Computing Applications.* Academic Press, San Diego, Ca.

[Murray, 1992] Murray, A. F., Baxter, D. J., Churcher, S., Hamilton, A., Reekie, H. M., and Tarassenko, L. (1992). The Edinburgh pulse stream implementation of a learning-oriented network (EPSILON) chip. In *Neural Information Processing Systems (NIPS) Conference.*

[Nehmzow, 1992] Nehmzow, U. (1992). *Experiments in Competence Acquisition for Autonomous Mobile Robots.* PhD thesis, University of Edinburgh.

[Widrow, 1988] Widrow, B. (1988). *DARPA Neural Network Study.* AFCEA International Press.